# Gaussian Process Dynamical Models

**Jack M. Wang, David J. Fleet, Aaron Hertzmann**
Department of Computer Science
University of Toronto, Toronto, ON M5S 3G4
{jmwang,hertzman}@dgp.toronto.edu, fleet@cs.toronto.edu

## Abstract

This paper introduces Gaussian Process Dynamical Models (GPDM) for nonlinear time series analysis. A GPDM comprises a low-dimensional latent space with associated dynamics, and a map from the latent space to an observation space. We marginalize out the model parameters in closed-form, using Gaussian Process (GP) priors for both the dynamics and the observation mappings. This results in a nonparametric model for dynamical systems that accounts for uncertainty in the model. We demonstrate the approach on human motion capture data in which each pose is 62-dimensional. Despite the use of small data sets, the GPDM learns an effective representation of the nonlinear dynamics in these spaces. **Webpage:** http://www.dgp.toronto.edu/~jmwang/gpdm/

## 1 Introduction

A central difficulty in modeling time-series data is in determining a model that can capture the nonlinearities of the data without overfitting. Linear autoregressive models require relatively few parameters and allow closed-form analysis, but can only model a limited range of systems. In contrast, existing nonlinear models can model complex dynamics, but may require large training sets to learn accurate MAP models.

In this paper we investigate learning nonlinear dynamical models for high-dimensional datasets. We take a Bayesian approach to modeling dynamics, averaging over dynamics parameters rather than estimating them. Inspired by the fact that averaging over nonlinear regression models leads to a Gaussian Process (GP) model, we show that integrating over parameters in nonlinear dynamical systems can also be performed in closed-form. The resulting Gaussian Process Dynamical Model (GPDM) is fully defined by a set of low-dimensional representations of the training data, with both dynamics and observation mappings learned from GP regression. As a natural consequence of GP regression, the GPDM removes the need to select many parameters associated with function approximators while retaining the expressiveness of nonlinear dynamics and observation.

Our work is motivated by modeling human motion for video-based people tracking and data-driven animation. Bayesian people tracking requires dynamical models in the form of transition densities in order to specify prediction distributions over new poses at each time instant (e.g., [11, 14]); similarly, data-driven computer animation requires prior distributions over poses and motion (e.g., [1, 4, 6]). An individual human pose is typically parameterized with more than 60 parameters. Despite the large state space, the space of activity-specific human poses and motions has a much smaller intrinsic dimensionality; in our experiments with walking and golf swings, 3 dimensions often suffice.

Our work builds on the extensive literature in nonlinear time-series analysis, of which we

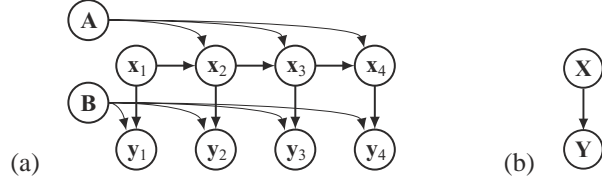

Figure 1: Time-series graphical models. (a) Nonlinear latent-variable model for time series. (Hyperparameters $\bar{\alpha}$ and $\bar{\beta}$ are not shown.) (b) GPDM model. Because the mapping parameters $\mathbf{A}$ and $\mathbf{B}$ have been marginalized over, all latent coordinates $\mathbf{X} = [\mathbf{x}_1, ..., \mathbf{x}_N]^T$ are jointly correlated, as are all poses $\mathbf{Y} = [\mathbf{y}_1, ..., \mathbf{y}_N]^T$.

mention a few examples. Two main themes are the use of switching linear models (e.g., [11]), and nonlinear transition functions, such as represented by Radial Basis Functions [2]. Both approaches require sufficient amounts of training data that one can learn the parameters of the switching or basis functions. Determining the appropriate number of basis functions is also difficult. In Kernel Dynamical Modeling [12], linear dynamics are kernelized to model nonlinear systems, but a density function over data is not produced.

Supervised learning with GP regression has been used to model dynamics for a variety of applications [3, 7, 13]. These methods model dynamics directly in observation space, which is impractical for the high-dimensionality of motion capture data. Our approach is most directly inspired by the unsupervised Gaussian Process Latent Variable Model (GPLVM) [5], which models the joint distribution of the observed data and their corresponding representation in a low dimensional latent space. This distribution can then be used as a prior for inference from new measurements. However, the GPLVM is not a dynamical model; it assumes that data are generated independently. Accordingly it does not respect temporal continuity of the data, nor does it model the dynamics in the latent space. Here we augment the GPLVM with a latent dynamical model. The result is a Bayesian generalization of subspace dynamical models to nonlinear latent mappings and dynamics.

## 2   Gaussian Process Dynamics

The Gaussian Process Dynamical Model (GPDM) comprises a mapping from a latent space to the data space, and a dynamical model in the latent space (Figure 1). These mappings are typically nonlinear. The GPDM is obtained by marginalizing out the parameters of the two mappings, and optimizing the latent coordinates of training data.

More precisely, our goal is to model the probability density of a sequence of vector-valued states $\mathbf{y}_1..., \mathbf{y}_t, ..., \mathbf{y}_N$, with discrete-time index $t$ and $\mathbf{y}_t \in \mathbb{R}^D$. As a basic model, consider a latent-variable mapping with first-order Markov dynamics:

$$\begin{aligned} \mathbf{x}_t &= f(\mathbf{x}_{t-1}; \mathbf{A}) + \mathbf{n}_{x,t} & (1)\\ \mathbf{y}_t &= g(\mathbf{x}_t; \mathbf{B}) + \mathbf{n}_{y,t} & (2) \end{aligned}$$

Here, $\mathbf{x}_t \in \mathbb{R}^d$ denotes the $d$-dimensional latent coordinates at time $t$, $\mathbf{n}_{x,t}$ and $\mathbf{n}_{y,t}$ are zero-mean, white Gaussian noise processes, $f$ and $g$ are (nonlinear) mappings parameterized by $\mathbf{A}$ and $\mathbf{B}$, respectively. Figure 1(a) depicts the graphical model.

While linear mappings have been used extensively in auto-regressive models, here we consider the nonlinear case for which $f$ and $g$ are linear combinations of basis functions:

$$\begin{aligned} f(\mathbf{x}; \mathbf{A}) &= \sum_i \mathbf{a}_i\, \phi_i(\mathbf{x}) & (3)\\ g(\mathbf{x}; \mathbf{B}) &= \sum_j \mathbf{b}_j\, \psi_j(\mathbf{x}) & (4) \end{aligned}$$

for weights $\mathbf{A} = [\mathbf{a}_1, \mathbf{a}_2, ...]$ and $\mathbf{B} = [\mathbf{b}_1, \mathbf{b}_2, ...]$, and basis functions $\phi_i$ and $\psi_j$. In order to fit the parameters of this model to training data, one must select an appropriate number of basis functions, and one must ensure that there is enough data to constrain the shape of each basis function. Ensuring both of these conditions can be very difficult in practice.

However, from a Bayesian perspective, the specific forms of $f$ and $g$ — including the numbers of basis functions — are incidental, and should therefore be marginalized out. With an isotropic Gaussian prior on the columns of $\mathbf{B}$, marginalizing over $g$ can be done in closed form [8, 10] to yield

$$p(\mathbf{Y} \,|\, \mathbf{X}, \bar{\beta}) \;=\; \frac{|\mathbf{W}|^N}{\sqrt{(2\pi)^{ND}|\mathbf{K}_Y|^D}} \exp\left( -\frac{1}{2}\mathrm{tr}\left( \mathbf{K}_Y^{-1} \mathbf{Y} \mathbf{W}^2 \mathbf{Y}^T \right) \right) \,, \qquad (5)$$

where $\mathbf{Y} = [\mathbf{y}_1, ..., \mathbf{y}_N]^T$, $\mathbf{K}_Y$ is a kernel matrix, and $\bar{\beta} = \{\beta_1, \beta_2, ..., \mathbf{W}\}$ comprises the kernel hyperparameters. The elements of kernel matrix are defined by a kernel function, $(\mathbf{K}_Y)_{i,j} = k_Y(\mathbf{x}_i, \mathbf{x}_j)$. For the latent mapping, $\mathbf{X} \to \mathbf{Y}$, we currently use the RBF kernel

$$k_Y(\mathbf{x}, \mathbf{x}') \;=\; \beta_1 \exp\left( -\frac{\beta_2}{2}||\mathbf{x} - \mathbf{x}'||^2 \right) + \beta_3^{-1} \delta_{\mathbf{x}, \mathbf{x}'} \,. \qquad (6)$$

As in the SGPLVM [4], we use a scaling matrix $\mathbf{W} \equiv \mathrm{diag}(w_1, ..., w_D)$ to account for different variances in the different data dimensions. This is equivalent to a GP with kernel function $k(\mathbf{x}, \mathbf{x}')/w_m^2$ for dimension $m$. Hyperparameter $\beta_1$ represents the overall scale of the output function, while $\beta_2$ corresponds to the inverse width of the RBFs. The variance of the noise term $\mathbf{n}_{y,t}$ is given by $\beta_3^{-1}$.

The dynamic mapping on the latent coordinates $\mathbf{X}$ is conceptually similar, but more subtle.[1] As above, we form the joint probability density over the latent coordinates and the dynamics weights $\mathbf{A}$ in (3). We then marginalize over the weights $\mathbf{A}$, i.e.,

$$p(\mathbf{X} \,|\, \bar{\alpha}) \;=\; \int p(\mathbf{X}, \mathbf{A} \,|\, \bar{\alpha})\, d\mathbf{A} \;=\; \int p(\mathbf{X} \,|\, \mathbf{A}, \bar{\alpha})\, p(\mathbf{A} \,|\, \bar{\alpha})\, d\mathbf{A} \,. \qquad (7)$$

Incorporating the Markov property (Eqn. (1)) gives:

$$p(\mathbf{X} \,|\, \bar{\alpha}) \;=\; p(\mathbf{x}_1) \int \prod_{t=2}^{N} p(\mathbf{x}_t \,|\, \mathbf{x}_{t-1}, \mathbf{A}, \bar{\alpha})\, p(\mathbf{A} \,|\, \bar{\alpha})\, d\mathbf{A} \,, \qquad (8)$$

where $\bar{\alpha}$ is a vector of kernel hyperparameters. Assuming an isotropic Gaussian prior on the columns of $\mathbf{A}$, it can be shown that this expression simplifies to:

$$p(\mathbf{X} \,|\, \bar{\alpha}) \;=\; p(\mathbf{x}_1) \frac{1}{\sqrt{(2\pi)^{(N-1)d}|\mathbf{K}_X|^d}} \exp\left( -\frac{1}{2}\mathrm{tr}\left( \mathbf{K}_X^{-1} \mathbf{X}_{out} \mathbf{X}_{out}^T \right) \right) \,, \qquad (9)$$

where $\mathbf{X}_{out} = [\mathbf{x}_2, ..., \mathbf{x}_N]^T$, $\mathbf{K}_X$ is the $(N-1) \times (N-1)$ kernel matrix constructed from $\{\mathbf{x}_1, ..., \mathbf{x}_{N-1}\}$, and $\mathbf{x}_1$ is assumed to be have an isotropic Gaussian prior.

We model dynamics using both the RBF kernel of the form of Eqn. (6), as well as the following "linear + RBF" kernel:

$$k_X(\mathbf{x}, \mathbf{x}') \;=\; \alpha_1 \exp\left( -\frac{\alpha_2}{2}||\mathbf{x} - \mathbf{x}'||^2 \right) + \alpha_3 \mathbf{x}^T \mathbf{x}' + \alpha_4^{-1} \delta_{\mathbf{x}, \mathbf{x}'} \,. \qquad (10)$$

The kernel corresponds to representing $g$ as the sum of a linear term and RBF terms. The inclusion of the linear term is motivated by the fact that linear dynamical models, such as

first or second-order autoregressive models, are useful for many systems. Hyperparameters $\alpha_1, \alpha_2$ represent the output scale and the inverse width of the RBF terms, and $\alpha_3$ represents the output scale of the linear term. Together, they control the relative weighting between the terms, while $\alpha_4^{-1}$ represents the variance of the noise term $\mathbf{n}_{x,t}$.

It should be noted that, due to the nonlinear dynamical mapping in (3), the joint distribution of the latent coordinates is *not* Gaussian. Moreover, while the density over the initial state may be Gaussian, it will not remain Gaussian once propagated through the dynamics. One can also see this in (9) since $\mathbf{x}_t$ variables occur inside the kernel matrix, as well as outside of it. So the log likelihood is not quadratic in $\mathbf{x}_t$.

Finally, we also place priors on the hyperparameters ($p(\bar{\alpha}) \propto \prod_i \alpha_i^{-1}$, and $p(\bar{\beta}) \propto \prod_i \beta_i^{-1}$) to discourage overfitting. Together, the priors, the latent mapping, and the dynamics define a generative model for time-series observations (Figure 1(b)):

$$p(\mathbf{X}, \mathbf{Y}, \bar{\alpha}, \bar{\beta}) \;=\; p(\mathbf{Y}|\mathbf{X}, \bar{\beta})\, p(\mathbf{X}|\bar{\alpha})\, p(\bar{\alpha})\, p(\bar{\beta}) \,. \tag{11}$$

**Multiple sequences.** This model extends naturally to multiple sequences $\mathbf{Y}_1, ..., \mathbf{Y}_M$. Each sequence has associated latent coordinates $\mathbf{X}_1, ..., \mathbf{X}_M$ within a shared latent space. For the latent mapping $g$ we can conceptually concatenate all sequences within the GP likelihood (Eqn. (5)). A similar concatenation applies for the dynamics, but omitting the first frame of each sequence from $\mathbf{X}_{out}$, and omitting the final frame of each sequence from the kernel matrix $\mathbf{K}_X$. The same structure applies whether we are learning from multiple sequences, or learning from one sequence and inferring another. That is, if we learn from a sequence $\mathbf{Y}_1$, and then infer the latent coordinates for a new sequence $\mathbf{Y}_2$, then the joint likelihood entails full kernel matrices $\mathbf{K}_X$ and $\mathbf{K}_Y$ formed from both sequences.

**Higher-order features.** The GPDM can be extended to model higher-order Markov chains, and to model velocity and acceleration in inputs and outputs. For example, a second-order dynamical model,

$$\mathbf{x}_t = f(\mathbf{x}_{t-1}, \mathbf{x}_{t-2}; \mathbf{A}) + \mathbf{n}_{x,t} \tag{12}$$

may be used to explicitly model the dependence of the prediction on two past frames (or on velocity). In the GPDM framework, the equivalent model entails defining the kernel function as a function of the current and previous time-step:

$$\begin{aligned} k_X\left([\mathbf{x}_t, \mathbf{x}_{t-1}], [\mathbf{x}_\tau, \mathbf{x}_{\tau-1}]\right) \;=\;\; & \alpha_1 \exp\left(-\frac{\alpha_2}{2}||\mathbf{x}_t - \mathbf{x}_\tau||^2 - \frac{\alpha_3}{2}||\mathbf{x}_{t-1} - \mathbf{x}_{\tau-1}||^2\right) \\ & + \alpha_4\, \mathbf{x}_t^T \mathbf{x}_\tau \;+\; \alpha_5\, \mathbf{x}_{t-1}^T \mathbf{x}_{\tau-1} \;+\; \alpha_6^{-1} \delta_{t,\tau} \end{aligned} \tag{13}$$

Similarly, the dynamics can be formulated to predict velocity:

$$\mathbf{v}_{t-1} = f(\mathbf{x}_{t-1}; \mathbf{A}) + \mathbf{n}_{x,t} \tag{14}$$

Velocity prediction may be more appropriate for modeling smoothly motion trajectories. Using Euler integration with time-step $\Delta t$, we have $\mathbf{x}_t = \mathbf{x}_{t-1} + \mathbf{v}_{t-1}\Delta t$. The dynamics likelihood $p(\mathbf{X}\,|\,\bar{\alpha})$ can then be written by redefining $\mathbf{X}_{out} = [\mathbf{x}_2 - \mathbf{x}_1, ..., \mathbf{x}_N - \mathbf{x}_{N-1}]^T/\Delta t$ in Eqn. (9). In this paper, we use a fixed time-step of $\Delta t = 1$. This is analogous to using $\mathbf{x}_{t-1}$ as a "mean function." Higher-order features can also be fused together with position information to reduce the Gaussian process prediction variance [15, 9].

## 3 Properties of the GPDM and Algorithms

Learning the GPDM from measurements $\mathbf{Y}$ entails minimizing the negative log-posterior:

$$\mathcal{L} \;=\; -\ln p(\mathbf{X}, \bar{\alpha}, \bar{\beta}\,|\,\mathbf{Y}) \tag{15}$$

$$= \frac{d}{2}\ln|\mathbf{K}_X| + \frac{1}{2}\mathrm{tr}\left(\mathbf{K}_X^{-1}\mathbf{X}_{out}\mathbf{X}_{out}^T\right) + \sum_j \ln\alpha_j \qquad (16)$$

$$- N\ln|\mathbf{W}| + \frac{D}{2}\ln|\mathbf{K}_Y| + \frac{1}{2}\mathrm{tr}\left(\mathbf{K}_Y^{-1}\mathbf{Y}\mathbf{W}^2\mathbf{Y}^T\right) + \sum_j \ln\beta_j$$

up to an additive constant. We minimize $\mathcal{L}$ with respect to $\mathbf{X}, \bar{\alpha}$, and $\bar{\beta}$ numerically.

Figure 2 shows a GPDM 3D latent space learned from a human motion capture data comprising three walk cycles. Each pose was defined by 56 Euler angles for joints, 3 global (torso) pose angles, and 3 global (torso) translational velocities. For learning, the data was mean-subtracted, and the latent coordinates were initialized with PCA. Finally, a GPDM is learned by minimizing $\mathcal{L}$ in (16). We used 3D latent spaces for all experiments shown here. Using 2D latent spaces leads to intersecting latent trajectories. This causes large "jumps" to appear in the model, leading to unreliable dynamics.

For comparison, Fig. 2(a) shows a 3D SGPLVM learned from walking data. Note that the latent trajectories are not smooth; there are numerous cases where consecutive poses in the walking sequence are relatively far apart in the latent space. By contrast, Fig. 2(b) shows that the GPDM produces a much smoother configuration of latent positions. Here the GPDM arranges the latent positions roughly in the shape of a saddle.

Figure 2(c) shows a volume visualization of the inverse reconstruction variance, i.e., $-2\ln\sigma_{\mathbf{y}|\mathbf{x},\mathbf{X},\mathbf{Y},\bar{\beta}}$. This shows the confidence with which the model reconstructs a pose from latent positions $\mathbf{x}$. In effect, the GPDM models a high probability "tube" around the data. To illustrate the dynamical process, Fig. 2(d) shows 25 fair samples from the latent dynamics of the GPDM. All samples are conditioned on the same initial state, $\mathbf{x}_0$, and each has a length of 60 time steps. As noted above, because we marginalize over the weights of the dynamic mapping, $\mathbf{A}$, the distribution over a pose sequence cannot be factored into a sequence of low-order Markov transitions (Fig. 1(a)). Hence, we draw fair samples $\tilde{\mathbf{X}}_{1:60}^{(j)} \sim p(\tilde{\mathbf{X}}_{1:60} \,|\, \mathbf{x}_0, \mathbf{X}, \mathbf{Y}, \bar{\alpha})$, using hybrid Monte Carlo [8]. The resulting trajectories (Fig. 2(c)) are smooth and similar to the training motions.

### 3.1 Mean Prediction Sequences

For both 3D people tracking and computer animation, it is desirable to generate new motions efficiently. Here we consider a simple online method for generating a new motion, called *mean-prediction*, which avoids the relatively expensive Monte Carlo sampling used above. In mean-prediction, we consider the next timestep $\tilde{\mathbf{x}}_t$ conditioned on $\tilde{\mathbf{x}}_{t-1}$ from the Gaussian prediction [8]:

$$\tilde{\mathbf{x}}_t \sim \mathcal{N}(\mu_X(\tilde{\mathbf{x}}_{t-1}); \sigma_X^2(\tilde{\mathbf{x}}_{t-1})\mathbf{I}) \qquad (17)$$

$$\mu_X(\mathbf{x}) = \mathbf{X}_{out}^T\mathbf{K}_X^{-1}\mathbf{k}_X(\mathbf{x}), \qquad \sigma_X^2(\mathbf{x}) = k_X(\mathbf{x}, \mathbf{x}) - \mathbf{k}_X(\mathbf{x})^T\mathbf{K}_X^{-1}\mathbf{k}_X(\mathbf{x}) \qquad (18)$$

where $\mathbf{k}_X(\mathbf{x})$ is a vector containing $k_X(\mathbf{x}, \mathbf{x}_i)$ in the $i$-th entry and $\mathbf{x}_i$ is the $i^{th}$ training vector. In particular, we set the latent position at each time-step to be the most-likely (mean) point given the previous step: $\tilde{\mathbf{x}}_t = \mu_X(\tilde{\mathbf{x}}_{t-1})$. In this way we ignore the process noise that one might normally add. We find that this mean-prediction often generates motions that are more like the fair samples shown in Fig. 2(d), than if random process noise had been added at each time step (as in (1)). Similarly, new poses are given by $\tilde{\mathbf{y}}_t = \mu_Y(\tilde{\mathbf{x}}_t)$.

Depending on the dataset and the choice of kernels, long sequences generated by sampling or mean-prediction can diverge from the data. On our data sets, mean-prediction trajectories from the GPDM with an RBF or linear+RBF kernel for dynamics usually produce sequences that roughly follow the training data (e.g., see the red curves in Figure 3). This usually means producing closed limit cycles with walking data. We also found that mean-prediction motions are often very close to the mean obtained from the HMC sampler; by

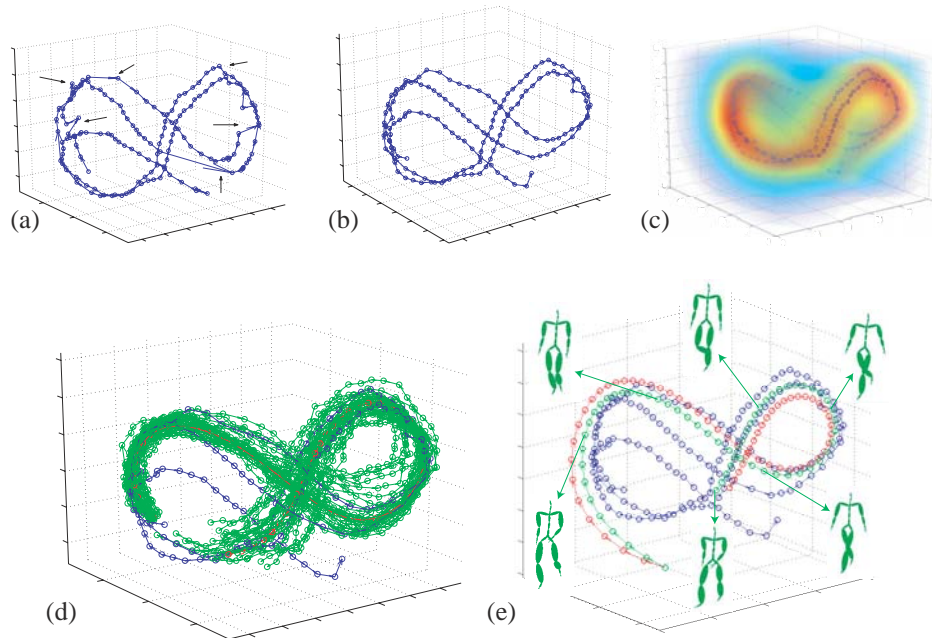

Figure 2: Models learned from a walking sequence of 2.5 gait cycles. The latent positions learned with a GPLVM (a) and a GPDM (b) are shown in blue. Vectors depict the temporal sequence. (c) - log variance for reconstruction shows regions of latent space that are reconstructed with high confidence. (d) Random trajectories drawn from the model using HMC (green), and their mean (red). (e) A GPDM of walk data learned with RBF+linear kernel dynamics. The simulation (red) was started far from the training data, and then optimized (green). The poses were reconstructed from points on the optimized trajectory.

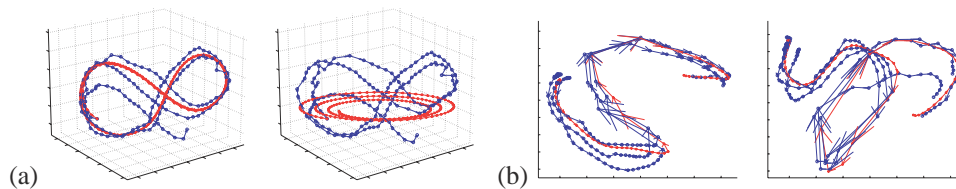

Figure 3: (a) Two GPDMs and mean predictions. The first is that from the previous figure. The second was learned with a linear kernel. (b) The GPDM model was learned from 3 swings of a golf club, using a $2^{nd}$ order RBF kernel for dynamics. The two plots show 2D orthogonal projections of the 3D latent space.

initializing HMC with mean-prediction, we find that the sampler reaches equilibrium in a small number of interations. Compared to the RBF kernels, mean-prediction motions generated from GPDMs with the linear kernel often deviate from the original data (e.g., see Figure 3a), and lead to over-smoothed animation.

Figure 3(b) shows a 3D GPDM learned from three swings of a golf club. The learning aligns the sequences and nicely accounts for variations in speed during the club trajectory.

## 3.2 Optimization

While mean-prediction is efficient, there is nothing in the algorithm that prevents trajectories from drifting away from the training data. Thus, it is sometimes desirable to optimize a particular motion under the GPDM, which often reduces drift of the mean-prediction mo-

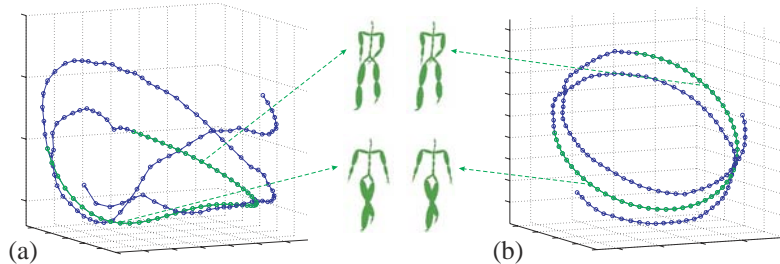

Figure 4: GPDM from walk sequence with missing data learned with (a) a RBF+linear kernel for dynamics, and (b) a linear kernel for dynamics. Blue curves depict original data. Green curves are the reconstructed, missing data.

tions. To optimize a new sequence, we first select a starting point $\tilde{\mathbf{x}}_1$ and a number of time-steps. The likelihood $p(\tilde{\mathbf{X}} \mid \mathbf{X}, \bar{\alpha})$ of the new sequence $\tilde{\mathbf{X}}$ is then optimized directly (holding the latent positions of the previously learned latent positions, $\mathbf{X}$, and hyperparameters, $\bar{\alpha}$, fixed). To see why optimization generates motion close to the traing data, note that the variance of pose $\tilde{\mathbf{x}}_{t+1}$ is determined by $\sigma_X^2(\tilde{\mathbf{x}}_t)$, which will be lower when $\tilde{\mathbf{x}}_t$ is nearer the training data. Consequently, the likelihood of $\tilde{\mathbf{x}}_{t+1}$ can be increased by moving $\tilde{\mathbf{x}}_t$ closer to the training data. This generalizes the preference of the SGPLVM for poses similar to the examples [4], and is a natural consequence of the Bayesian approach. As an example, Fig. 2(e) shows an optimized walk sequence initialized from the mean-prediction.

### 3.3 Forecasting

We performed a simple experiment to compare the predictive power of the GPDM to a linear dynamical system, implemented as a GPDM with linear kernel in the latent space and RBF latent mapping. We trained each model on the first 130 frames of the 60Hz walking sequence (corresponding to 2 cycles), and tested on the remaining 23 frames. From each test frame mean-prediction was used to predict the pose 8 frames ahead, and then the RMS pose error was computed against ground truth. The test was repeated using mean-prediction and optimization for three kernels, all based on first-order predictions as in (1):

|  | Linear | RBF | Linear+RBF |
|---|---|---|---|
| mean-prediction | 59.69 | 48.72 | 36.74 |
| optimization | 58.32 | 45.89 | 31.97 |

Due to the nonlinear nature of the walking dynamics in latent space, the RBF and Linear+RBF kernels outperform the linear kernel. Moreover, optimization (initialized by mean-prediction) improves the result in all cases, for reasons explained above.

### 3.4 Missing Data

The GPDM model can also handle incomplete data (a common problem with human motion capture sequences). The GPDM is learned by minimizing $\mathcal{L}$ (Eqn. (16)), but with the terms corresponding to missing poses $\mathbf{y}_t$ removed. The latent coordinates for missing data are initialized by cubic spline interpolation from the 3D PCA initialization of observations.

While this produces good results for short missing segments (e.g., 10–15 frames of the 157-frame walk sequence used in Fig. 2), it fails on long missing segments. The problem lies with the difficulty in initializing the missing latent positions sufficiently close to the training data. To solve the problem, we first learn a model with a subsampled data sequence. Reducing sampling density effectively increases uncertainty in the reconstruction process so that the probability density over the latent space falls off more smoothly from the data. We then restart the learning with the entire data set, but with the kernel hyperparameters fixed. In doing so, the dynamics terms in the objective function exert more influence over the latent coordinates of the training data, and a smooth model is learned.

With 50 missing frames of the 157-frame walk sequence, this optimization produces mod-

els (Fig. 4) that are much smoother than those in Fig. 2. The linear kernel is able to pull the latent coordinates onto a cylinder (Fig. 4b), and thereby provides an accurate dynamical model. Both models shown in Fig. 4 produce estimates of the missing poses that are visually indistinguishable from the ground truth.

## 4 Discussion and Extensions

One of the main strengths of the GPDM model is the ability to generalize well from small datasets. Conversely, performance is a major issue in applying GP methods to larger datasets. Previous approaches prune uninformative vectors from the training data [5]. This is not straightforward when learning a GPDM, however, because each timestep is highly correlated with the steps before and after it. For example, if we hold $\mathbf{x}_t$ fixed during optimization, then it is unlikely that the optimizer will make much adjustment to $\mathbf{x}_{t+1}$ or $\mathbf{x}_{t-1}$. The use of higher-order features provides a possible solution to this problem. Specifically, consider a dynamical model of the form $\mathbf{v}_t = f(\mathbf{x}_{t-1}, \mathbf{v}_{t-1})$. Since adjacent time-steps are related only by the velocity $\mathbf{v}_t \approx (\mathbf{x}_t - \mathbf{x}_{t-1})/\Delta t$, we can handle irregularly-sampled datapoints by adjusting the timestep $\Delta t$, possibly using a different $\Delta t$ at each step.

A number of further extensions to the GPDM model are possible. It would be straightforward to include a control signal $\mathbf{u}_t$ in the dynamics $f(\mathbf{x}_t, \mathbf{u}_t)$. It would also be interesting to explore uncertainty in latent variable estimation (e.g., see [3]). Our use of maximum likelihood latent coordinates is motivated by Lawrence's observation that model uncertainty and latent coordinate uncertainty are interchangeable when learning PCA [5]. However, in some applications, uncertainty about latent coordinates may be highly structured (e.g., due to depth ambiguities in motion tracking).

**Acknowledgements** This work made use of Neil Lawrence's publicly-available GPLVM code, the CMU mocap database (mocap.cs.cmu.edu), and Joe Conti's volume visualization code from mathworks.com. This research was supported by NSERC and CIAR.

## Footnotes

[1]Conceptually, we would like to model each pair $(\mathbf{x}_t, \mathbf{x}_{t+1})$ as a training pair for regression with $g$. However, we cannot simply substitute them directly into the GP model of Eqn. (5) as this leads to the nonsensical expression $p(\mathbf{x}_2, ..., \mathbf{x}_N \,|\, \mathbf{x}_1, ..., \mathbf{x}_{N-1})$.

## References

[1] M. Brand and A. Hertzmann. Style machines. *Proc. SIGGRAPH*, pp. 183-192, July 2000.
[2] Z. Ghahramani and S. T. Roweis. Learning nonlinear dynamical systems using an EM algorithm. *Proc. NIPS 11*, pp. 431-437, 1999.
[3] A. Girard, C. E. Rasmussen, J. G. Candela, and R. Murray-Smith. Gaussian process priors with uncertain inputs - application to multiple-step ahead time series forecasting. *Proc. NIPS 15*, pp. 529-536, 2003.
[4] K. Grochow, S. L. Martin, A. Hertzmann, and Z. Popović. Style-based inverse kinematics. *ACM Trans. Graphics*, 23(3):522-531, Aug. 2004.
[5] N. D. Lawrence. Gaussian process latent variable models for visualisation of high dimensional data. *Proc. NIPS 16*, 2004.
[6] J. Lee, J. Chai, P. S. A. Reitsma, J. K. Hodgins, and N. S. Pollard. Interactive control of avatars animated with human motion data. *ACM Trans. Graphics*, 21(3):491-500, July 2002.
[7] W. E. Leithead, E. Solak, and D. J. Leith. Direct identification of nonlinear structure using Gaussian process prior models. *Proc. European Control Conference*, 2003.
[8] D. MacKay. *Information Theory, Inference, and Learning Algorithms*. 2003.
[9] R. Murray-Smith and B. A. Pearlmutter. Transformations of Gaussian process priors. Technical Report, Department of Computer Science, Glasgow University, 2003
[10] R. M. Neal. *Bayesian Learning for Neural Networks*. Springer-Verlag, 1996.
[11] V. Pavlović, J. M. Rehg, and J. MacCormick. Learning switching linear models of human motion. *Proc. NIPS 13*, pp. 981-987, 2001.
[12] L. Ralaivola and F. d'Alché-Buc. Dynamical modeling with kernels for nonlinear time series prediction. *Proc. NIPS 16*, 2004.
[13] C. E. Rasmussen and M. Kuss. Gaussian processes in reinforcement learning. *Proc. NIPS 16*, 2004.
[14] H. Sidenbladh, M. J. Black, and D. J. Fleet. Stochastic tracking of 3D human figures using 2D motion. *Proc. ECCV*, volume 2, pp. 702-718, 2000.
[15] E. Solak, R. Murray-Smith, W. Leithead, D. Leith, and C. E. Rasmussen. Derivative observations in Gaussian process models of dynamic systems. *Proc. NIPS 15*, pp. 1033-1040, 2003.